# Implicitly Constrained Gaussian Process Regression for Monocular Non-Rigid Pose Estimation

**Mathieu Salzmann**
ICSI & EECS, UC Berkeley
TTI Chicago
salzmann@ttic.edu

**Raquel Urtasun**
TTI Chicago
rurtasun@ttic.edu

## Abstract

Estimating 3D pose from monocular images is a highly ambiguous problem. Physical constraints can be exploited to restrict the space of feasible configurations. In this paper we propose an approach to constraining the prediction of a discriminative predictor. We first show that the mean prediction of a Gaussian process implicitly satisfies linear constraints if those constraints are satisfied by the training examples. We then show how, by performing a change of variables, a GP can be forced to satisfy quadratic constraints. As evidenced by the experiments, our method outperforms state-of-the-art approaches on the tasks of rigid and non-rigid pose estimation.

## 1 Introduction

Estimating the 3D pose of an articulated body or of a deformable surface from monocular images is one of the fundamental problems in computer vision. It is known to be highly ambiguous and therefore requires the use of prior knowledge to restrict the pose to feasible configurations. Throughout the years, two main research directions have emerged to provide such knowledge: approaches that rely on modeling the explicit properties of the object of interest, and techniques that learn these properties from data.

Methods that exploit known physical properties have been proposed both for deformable shape recovery [9, 15] and for articulated pose estimation [18, 7]. Unfortunately, in most cases, the constraints introduced to disambiguate the pose are quadratic and non-convex. This, for example, is the case of fixed-length constraints [9, 15, 18] or unit norm constraints. As a consequence, such constraints are hard to optimize and often yield solutions that are sub-optimal.

Learning a prior over possible poses seems an attractive alternative [13, 3, 16, 22]. However, these priors are employed in generative approaches that require accurate initialization. In articulated pose estimation [14, 1, 21], the need for initialization has often been prevented by relying on discriminative predictors that learn a mapping from image observations to 3D pose. Unfortunately, the employed approaches typically assume that the output dimensions are independent given the inputs and are therefore only adapted to cases where the outputs are weakly correlated. In pose estimation, this independence assumption is in general violated, and these techniques yield solutions that are far from optimal. Recently, [12] proposed to overcome this issue by imposing additional physical constraints on the pose estimated by the predictor. However, these constraints were imposed at inference, which required to solve a non-convex optimization problem for each test example.

In this paper, we propose to make the predictor implicitly satisfy physical constraints. This lets us overcome the issues related to the output independence assumption of discriminative methods while avoiding the computational burden of enforcing constraints at inference. To this end, we first show that the mean prediction of a Gaussian process implicitly satisfies linear constraints. We then address the case of quadratic constraints by replacing the original unknowns of our problem with quadratic unknowns under which the constraints are linear. We demonstrate the effectiveness of our approach to predict rotations expressed either as quaternions under unit $L2$-norm constraints,

or as rotation matrices under orthonormality constraints, as well as to predict 3D non-rigid poses under constant length constraints. Our experiments show that our approach significantly outperforms Gaussian process regression, as well as imposing constraints at inference [12]. Furthermore, for high dimensional inputs and large training sets, our approach is orders of magnitude faster than [12].

## 2   Constrained Gaussian Process Regression

In this section, we first review Gaussian process regression and then show that, for vector outputs, linear constraints between the output dimensions are implicitly satisfied by the predictor. We then propose a change in parameterization that enables us to incorporate quadratic constraints. Finally, we rely on a simple factorization to recover the variables of interest for the pose estimation problem.

### 2.1   Gaussian Process Regression

A Gaussian process is a collection of random variables, any finite number of which have consistent joint Gaussian distributions [10]. Let $\mathcal{D} = \{(\mathbf{x}_i, y_i), i = 1, \cdots, N\}$ be a training set composed of inputs $\mathbf{x}_i$ and noisy outputs $y_i$ generated from a latent function $f(\mathbf{x})$ with i.i.d. Gaussian noise $\epsilon_i \sim \mathcal{N}(0, \sigma_n^2)$, such that $y_i = f(\mathbf{x}_i) + \epsilon_i$. Let $\mathbf{f} = [f(\mathbf{x}_1), \cdots, f(\mathbf{x}_N)]^T$ be the vector of function values, and $\mathbf{X} = [\mathbf{x}_1, \cdots, \mathbf{x}_N]^T$ be the inputs. GP regression assumes a GP prior over functions,

$$p(\mathbf{f}|\mathbf{X}) = \mathcal{N}(0, \mathbf{K}) , \tag{1}$$

where $\mathbf{K}$ is a covariance matrix whose entries are given by a covariance function, $K_{i,j} = k(\mathbf{x}_i, \mathbf{x}_j)$. Inference in the GP model is straightforward. Let $\mathbf{y} = [y_1, \cdots, y_N]^T$ be the vector of training outputs. Given a new input $\mathbf{x}_*$, the prediction $f(\mathbf{x}_*)$ follows a Gaussian distribution with mean $\mu(\mathbf{x}_*) = \mathbf{y}^T \mathbf{K}^{-1} \mathbf{k}_*$ and variance $\sigma(\mathbf{x}_*) = k(\mathbf{x}_*, \mathbf{x}_*) - \mathbf{k}_*^T \mathbf{K}^{-1} \mathbf{k}_*$.

The simplest way to extend a GP to deal with multiple outputs is to assume that, given the inputs, the outputs are independent. However, for correlated output dimensions, this is a poor approximation. Recent research has focused on learning the interactions between the output dimensions [6, 19, 4, 2]. In this paper, we take an alternative approach, since, for the problem at hand, the constraints are known a priori. In particular, we show that for any input we can enforce the mean prediction of a GP to implicitly satisfy the constraints linking the output dimensions, and thus implicitly model the output correlations.

### 2.2   Linear Constraints

As mentioned above, we consider the problem of vector output regression and seek to learn a predictor able to model the constraints linking the different dimensions of the output. Let us first study the case of linear relationships between the output dimensions. Here, we show that, if the training data satisfies a set of linear constraints, the mean prediction of a GP implicitly satisfies these constraints.

**Proposition 1.** *Let* $\{\mathbf{y}_1, \cdots, \mathbf{y}_N\}$ *be a set of training examples that satisfy the linear constraints* $\mathbf{A}\mathbf{y}_i = \mathbf{b}$. *For any input* $\mathbf{x}_*$, *the mean prediction of a Gaussian process* $\boldsymbol{\mu}(\mathbf{x}_*)$ *will also satisfy the constraints* $\mathbf{A}\boldsymbol{\mu}(\mathbf{x}_*) = \mathbf{b}$.

**Proof:** Let $\hat{\mathbf{Y}} = [\hat{\mathbf{y}}_1, \cdots, \hat{\mathbf{y}}_N]^T$ be the matrix of mean-subtracted training examples. The prediction of a GP can be computed as the sum of the mean of the training data, $\bar{\mathbf{y}}$, and a linear combination of the $\hat{\mathbf{y}}_i$, i.e., $\boldsymbol{\mu}(\mathbf{x}_*) = \bar{\mathbf{y}} + \hat{\mathbf{Y}}^T \mathbf{K}^{-1} \mathbf{k}_*$. The mean of the training data satisfies the constraints since $\mathbf{A}\bar{\mathbf{y}} = \mathbf{A}(\frac{1}{N} \sum_{i=1}^N \mathbf{y}_i) = \frac{1}{N} \sum_{i=1}^N \mathbf{b} = \mathbf{b}$. Furthermore, any mean-subtracted training example $\hat{\mathbf{y}}_i$ satisfies $\mathbf{A}\hat{\mathbf{y}}_i = 0$, since $\mathbf{A}\hat{\mathbf{y}}_i = \mathbf{A}\mathbf{y}_i - \mathbf{A}\bar{\mathbf{y}} = \mathbf{b} - \mathbf{b} = 0$. As a consequence, any linear combination $\hat{\mathbf{Y}}^T \mathbf{w}$ of the mean-subtracted training data satisfies $\mathbf{A}\hat{\mathbf{Y}}^T \mathbf{w} = \sum_i \mathbf{w}_i \mathbf{A}\hat{\mathbf{y}}_i = 0$. This, in particular, is the case when $\mathbf{w} = \mathbf{K}^{-1} \mathbf{k}_*$, and therefore $\mathbf{A}\boldsymbol{\mu}(\mathbf{x}_*) = \mathbf{A}(\bar{\mathbf{y}} + \hat{\mathbf{Y}}^T \mathbf{K}^{-1} \mathbf{k}_*) = \mathbf{b}$. $\square$

Thus, if the training examples satisfy linear constraints, the mean prediction of the GP will always satisfy the constraints. Note that this result not only holds for GPs, but for any predictor whose output is a linear combination of the training outputs. However, most of the physical constraints that rule non-rigid motion are more complex than simple linear functions. In particular, quadratic constraints are commonly used [9, 15].

### 2.3   Quadratic Constraints

We now show how we can enforce the prediction to satisfy quadratic constraints. To this end, we propose to perform a change in parameterization such that the constraints become linear in the new

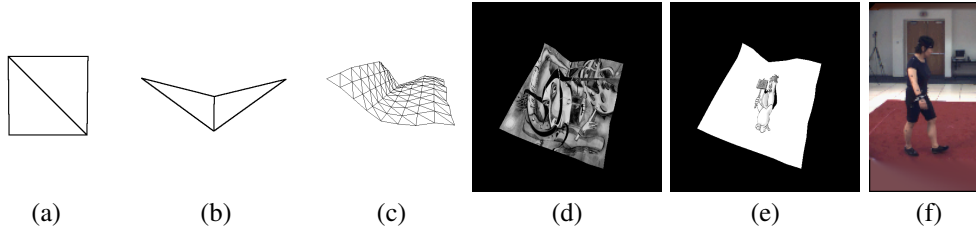

| (a) | (b) | (c) | (d) | (e) | (f) |

Figure 1: **Samples from our datasets.** (a) Square plane used for rotation estimation. (b) Similar square deformed by assigning a random value to the angle between its facets. (c) Synthetically generated inextensible mesh. (d) Image generated by texturing the mesh in (c). (e) Similar image obtained with a more uniform texture. (f) Image from the HumanEva dataset [17] registered with the method of [11].

variables. This can simply be achieved by replacing the original variables with their pairwise products. In cases where we do not expect the constraints to depend on all the quadratic variables (e.g., distance constraints between 3D points), we can consider only a subset of them. In this paper, we will investigate three types of quadratic constraints involved in rigid and non-rigid pose estimation.

More formally, let $\mathbf{Z} \in \Re^{P \times D}$ be a matrix encoding a training point. For rotations, when expressed in quaternion space, $P = 1$ and $D = 4$, and when expressed as rotation matrices, $P = 1$ and $D = 9$. In the case of a non-rigid pose expressed as a set of 3 dimensional points (e.g., human joints or mesh vertices), $P = 3$, and $D$ is the number of points representing the pose. Let $\mathbf{Q} \in \Re^{D \times D}$ be the matrix encoding quadratic variables such that $\mathbf{Q} = \mathbf{Z}^T \mathbf{Z}$. Since by definition $\mathbf{Q}$ is symmetric, it is fully determined by its upper triangular part. Thus, we define a training point for our Gaussian process as the concatenation of the upper triangular elements of $\mathbf{Q}$, i.e.,

$$\mathbf{y} = [\mathbf{Q}_{11}, \cdots, \mathbf{Q}_{ij}, \cdots \mathbf{Q}_{DD}]^T \qquad \text{with} \qquad i \geq j.$$

Note that, when $P = 1$, any quadratic equality constraint can be written as a linear equality constraint in terms of $\mathbf{y}$. As shown in the previous section, the prediction of a Gaussian process will always satisfy linear constraints if the training points satisfy the constraints. As a consequence, the variables we regress to will satisfy the quadratic constraints, and by construction, the matrix $\tilde{\mathbf{Q}}$ built from the mean prediction of the GP will be symmetric.

However, to solve the pose estimation problem, we are interested in $\mathbf{Z}$, not in $\mathbf{y}$. Thus, we need an additional step that transforms the quadratic variables into the original variables. We propose to cast this problem as a matrix factorization problem and minimize the Frobenius norm between the factorization and the output of the GP. The solution to this problem can be obtained in closed form by computing the SVD of the matrix built from the predicted quadratic variables $\tilde{\mathbf{y}} = \mu(\mathbf{x}_*)$, i.e.,

$$\tilde{\mathbf{Q}} = \begin{pmatrix} \tilde{y}_1 & \tilde{y}_2 & \cdots & \tilde{y}_D \\ \tilde{y}_2 & \cdots & \cdots & \cdots \\ \vdots & \vdots & \vdots & \vdots \\ \tilde{y}_D & \cdots & \cdots & \tilde{y}_{\frac{D(D+1)}{2}} \end{pmatrix} = \mathbf{V}\Sigma\mathbf{V}^T .$$

The final solution is obtained by taking into account only the singular vectors corresponding to the $P$ largest singular values. Assuming that the values in $\Sigma$ are ordered, this yields

$$\tilde{\mathbf{Z}} = \sqrt{\Sigma_{1:P,1:P}} \mathbf{V}_{:,1:P}^T , \tag{2}$$

where the subscript $1 : P$ denotes the first $P$ rows or columns of a matrix. Note that the GP does not guarantee that the predicted $\tilde{\mathbf{Q}}$ has rank $P$. Therefore, we do not truly guarantee that $\tilde{\mathbf{Z}}$ satisfies the constraints. However, as shown in our experiments, the violation of the constraints induced by the factorization is much smaller than the one produced by doing prediction in the original variables.

Note that the solution to the factorization of Eq. 2 is not unique. First, it is subject to $P$ sign ambiguities arising from taking the square root of $\Sigma$. Second, when $P > 1$, the solution can only be determined up to an orthonormal transformation $\mathbf{T}$, since $(\mathbf{V}_{:,1:P}\mathbf{T})\Sigma_{1:P,1:P}(\mathbf{V}_{:,1:P}\mathbf{T})^T = (\mathbf{V}_{:,1:P}\mathbf{T})\Sigma_{1:P,1:P}(\mathbf{T}^T\mathbf{V}_{:,1:P}^T) = \mathbf{V}_{:,1:P}\Sigma_{1:P,1:P}\mathbf{V}_{:,1:P}^T$. To overcome both sources of ambiguities, we rely on image information. Since we consider the case of rigid and non-rigid pose estimation, we can make the typical assumption that we have correspondences between 3D points on the object of interest and 2D image locations [8, 9]. The sign ambiguities result in $2^P$ discrete solutions. We disambiguate between them by choosing the one that yields the smallest reprojection

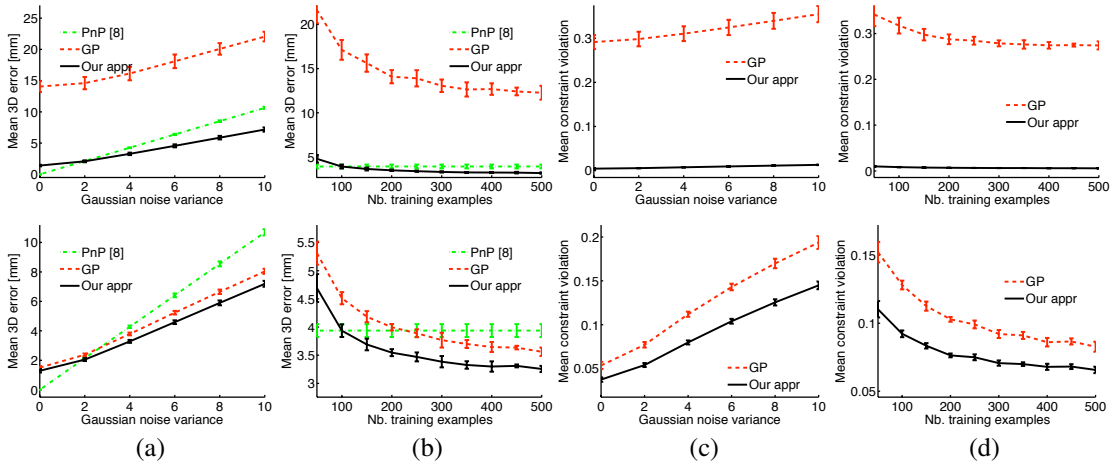

Figure 2: **Estimating the rotation of a plane.** (Top) Mean reconstruction error and constraint violation when parameterizing the rotations with quaternions. (Bottom) Similar plots when the rotations were parameterized as rotation matrices. Note that our approach outperforms the baselines and is insensitive to the parameterization used.

error. Note, however, that other types of image information, such as silhouettes or texture, could also be employed. To determine the global transformation of $\tilde{\mathbf{Z}}$, we similarly rely on 3D-to-2D correspondences; finding a rigid transformation that minimizes the reprojection error of 3D points is a well-studied problem in computer vision, called the *PnP problem*. In practice, we employ the closed-form solution of [8] to estimate $\mathbf{T}$.

## 3 Experimental Evaluation

In this section, we show our results on rigid and non-rigid reconstruction problems involving quadratic constraints. Samples from the diverse datasets employed are depicted in Fig. 1. As our error measure, we report mean point-to-point distance between the recovered 3D shape and ground-truth averaged over 10 partitions for a fixed test set size of 500 examples. Furthermore, we also show error bars that represent $\pm$ one standard deviation computed over the 10 partitions. These error bars are non-overlapping for all constraint violation plots, as well as for most of the reconstruction errors, which shows that our results are statistically significant. For all experiments we used a covariance function which is the sum of an RBF and a noise term, and fixed the width of the RBF to the mean squared distance between the training inputs and the noise variance to $\sigma_n^2 = 0.01$. Furthermore, in cases where the number of training examples is smaller than the output dimensionality (i.e. for large deformable meshes and for human poses), we performed principal component analysis on the training outputs to speed up training. To entail no loss in the data, we only removed the components with corresponding zero-valued eigenvalues.

### 3.1 Rotation of a Plane

First, we considered the case of inferring the rotation in 3D space of the square in Fig. 1(a) given noisy 2D image observations of its corners. Note that this is an instance of the PnP problem. We used two different parameterizations of the rotations: quaternions and rotation matrices. In the first case, the recovered quaternion must have unit norm, i.e., $||\tilde{\mathbf{Z}}||_2 = 1$. In the second case, the recovered rotation matrix must be orthonormal, i.e., $\tilde{\mathbf{Z}}^T \tilde{\mathbf{Z}} = \mathbf{I}$.

Fig. 2(a,b) depicts the reconstruction errors obtained with quaternions (top) and rotation matrices (bottom), as a function of the Gaussian noise variance on the 2D image locations when using a training set of 100 examples (a), and as a function of the number of training examples for a Gaussian noise variance of 5 (b). We compare the results of our approach to those obtained by a GP trained on the original variables, as well as to the results of a state-of-the-art PnP method [8], which would be the standard approach to solving this problem. In all cases, our approach outperforms the baselines. More importantly, our approach performs equally well for all the parameterizations of the rotation.

Fig. 2(c,d) shows the mean constraint violation for both parameterizations. For quaternions, this error is computed as the absolute difference between the norm of the recovered quaternion and 1. For rotation matrices it is computed as the Frobenius norm of the difference between $\tilde{\mathbf{Z}}^T \tilde{\mathbf{Z}}$ and

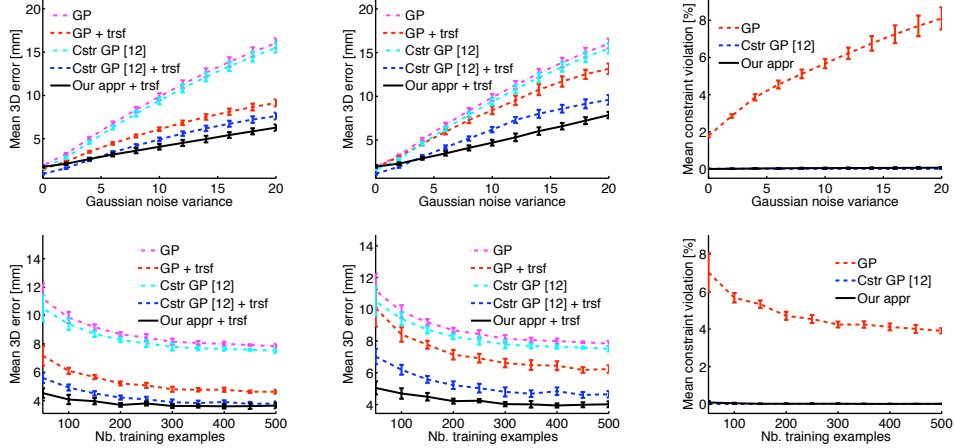

Figure 3: **Estimating the 3D shape of a** $2 \times 2$ **mesh from 2D image locations.** (Top) Mean reconstruction error and constraint violation as a function of the input noise. The global transformation was estimated either (left) from the ground truth, or (middle) using a PnP method [8]. (Bottom) Similar errors shown as a function of the number of training examples.

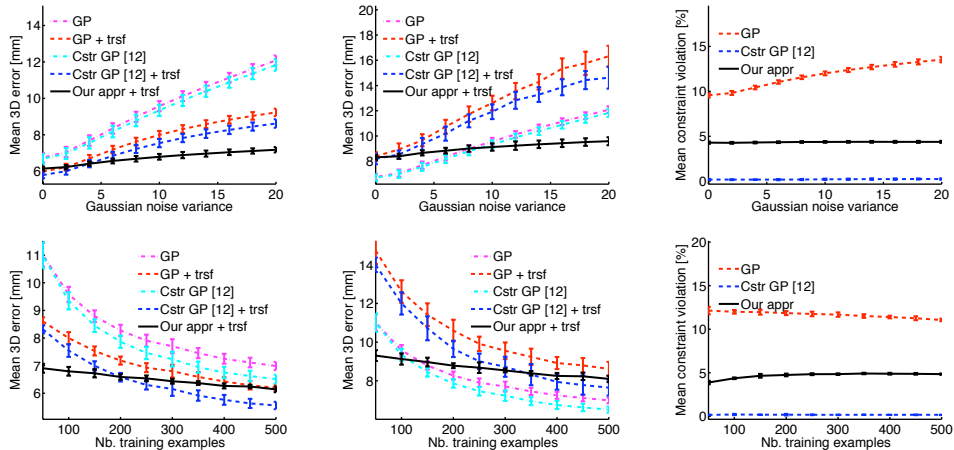

Figure 4: **Estimating the 3D shape of a** $9 \times 9$ **mesh from 2D image locations.** (Top) Mean reconstruction error and constraint violation as a function of the input noise. The global transformation was estimated either (left) from the ground truth, or (middle) using a PnP method [8]. (Bottom) Similar errors shown as a function of the number of training examples. Note that the global transformations estimated with the PnP method yield poor reconstructions. However, our approach performs best among those that use these transformations.

the identity matrix. Note that in both cases, our approach better satisfies the quadratic constraints than the standard GP. This is especially true in the case of unit norm quaternions, where the results obtained with the GP strongly violate the constraints.

### 3.2 Surface Deformations

Next, we considered the problem of estimating the shape of a deforming surface from a single image. In this context, the output space is composed of the 3D locations of the vertices of the mesh that represents the surface, and the quadratic constraints encode the fact that the length of the mesh edges should remain constant. The constraint error measure was taken to be the average over all edges of the percentage of length variation. We compare against two baselines, GP in the original variables (i.e., 3D locations of mesh vertices), and the approach of [12] where the constraints are explicitly enforced at inference. Since our approach only allows us to recover the shape up to a global transformation, we show results estimating this transformation either from the ground-truth data, which can be done by computing an SVD [20], or by applying a PnP method [8]. To make our evaluation fair, we also computed similar global transformations for the baselines.

We tested our approach on the same square as before, but allowing it to deform by letting the edge between its two facets act as a hinge, as shown in Fig. 1(b). Doing so ensures that the length of

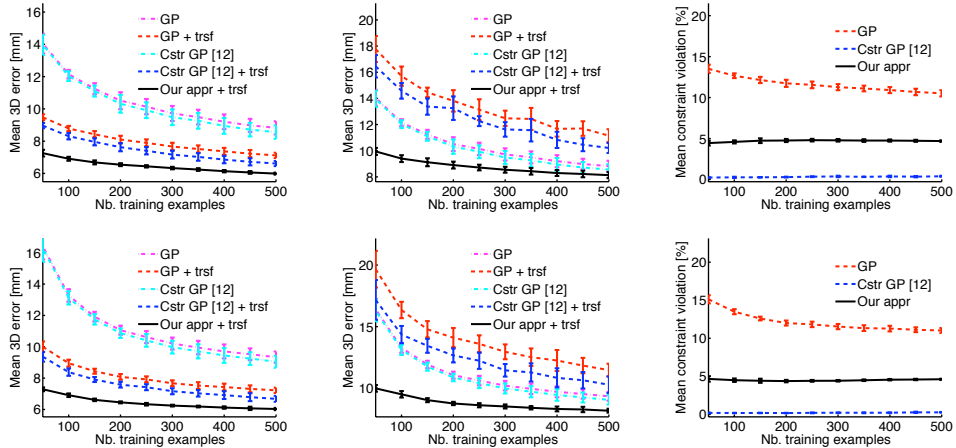

Figure 5: **Estimating the 3D shape of a** $9 \times 9$ **mesh from PHOG features.** Mean reconstruction error and constraint violation obtained from (top) well-textured images (Fig. 1(d)), or (bottom) poorly-textured ones (Fig. 1(e)). The global transformation was estimated either (left) from the ground truth, or (middle) using a PnP method [8].

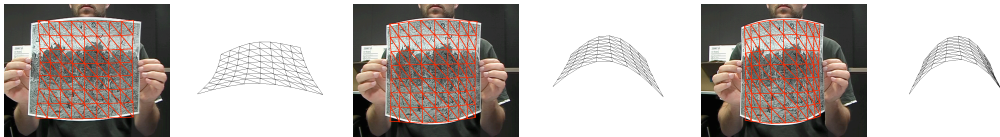

Figure 6: **Non-rigid reconstruction from real images.** Reconstructions of a piece of paper from 2D image locations. We show the recovered mesh overlaid in red on the original image, and a side view of this mesh.

the mesh edges remains constant. Similarly as before, the inputs to the GP, $\mathbf{x}$, were taken to be the 2D image locations of the corners of the square. Fig. 3 depicts the reconstruction error and constraint violation as a function of the Gaussian noise variance added to the 2D image locations for training sets composed of 100 training examples (top), and as a function of the number of training examples for a Gaussian noise variance of 10 (bottom). Note that our approach is more robust to input noise than the baselines. Furthermore, unlike the standard GP, our approach satisfies the quadratic constraints.

We then tested our approach on the larger mesh shown in Fig. 1(c). In that case, the matrix $\mathbf{Z} \in \Re^{3 \times 81}$. We generated inextensible deformed mesh examples by randomly sampling the values of a subset of the angles between the facets of the mesh. Fig. 4 depicts the results obtained when using the 2D image locations as inputs. As before, we can see that our approach is more robust to input noise than the baselines[1]. Note that the global transformations estimated with the PnP method tend to be inaccurate and therefore yield poor results. However, our approach performs best among the ones that utilize the PnP method. We can also notice that our approach better satisfies the constraints than GP prediction in the original space. The small violation of the constraints is due to the fact that our prediction is not guaranteed to be rank 3, and therefore the factorization may result in some loss. We then considered the more general case of having images as inputs instead of the 2D locations of the mesh vertices. For this purpose, we generated images such as those of Fig. 1(d,e) from which we computed PHOG features [5]. As shown in Fig. 5, our approach outperforms the baselines for all training set sizes.

To demonstrate our method's ability to deal with real images, we reconstructed the deformations of a piece of paper from a video sequence. We used the 2D image locations of the vertices of the mesh as inputs, which were obtained by tracking the surface in 2D using template matching. For this case, the training data was obtained by deforming a piece of cardboard in front of an optical motion capture system. Results for some frames of the sequence are shown in Fig. 6. Note that, for small deformations, the problem is subject to concave-convex ambiguities arising from the insufficient perspective. As a consequence, the shape is less accurate than when the deformations are larger.

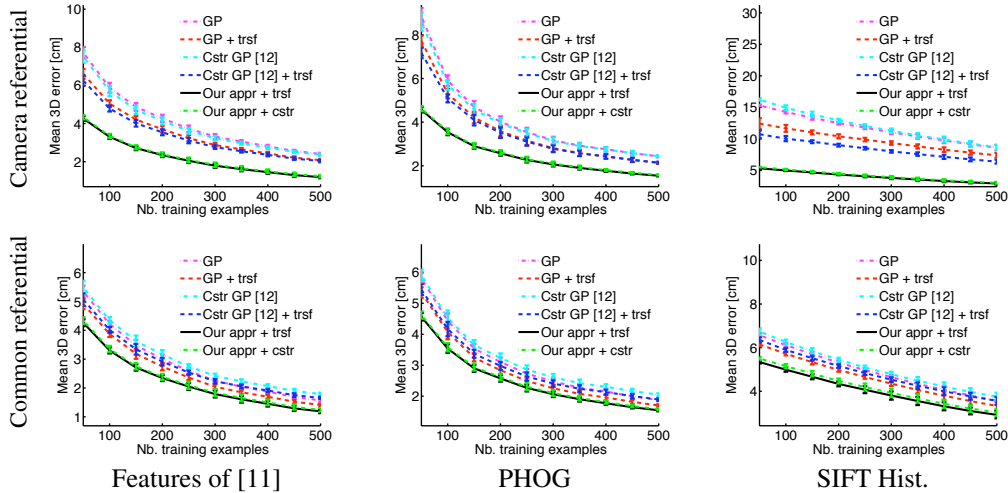

Figure 7: **Human pose estimation from different image features.** Mean reconstruction error as a function of the number of training examples for 3 different feature types and with the pose represented in 2 different referentials.

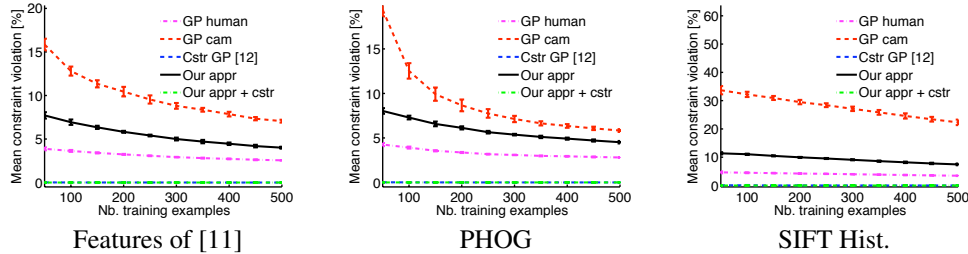

Figure 8: **Constraint violation in human pose estimation.** Mean constraint violation for 3 different image feature types. Note that the constrained GP [12] best satisfies the constraints, since it explicitly enforces them at inference. However, our approach is more stable than the standard GP.

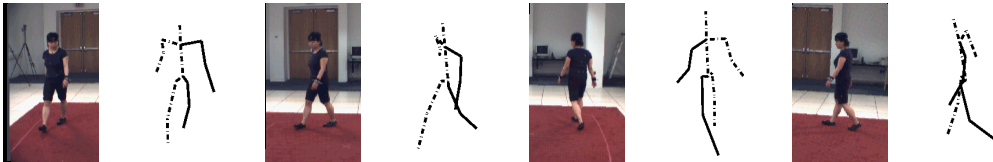

Figure 9: **Human pose estimation from real images.** We show the rectified image from [11] and the pose recovered by our approach using PHOG features as input seen from a different viewpoint.

### 3.3 Human Pose Estimation

We also applied our method to the problem of estimating the pose of a human skeleton. To this end, we used the HumanEva dataset [17], which consists of synchronized images and motion capture data. In particular, we used the rectified images of [11] and relied on three different image features as input: histograms of SIFT features, PHOG features, and the features of [11]. In this case $\mathbf{Z} \in \Re^{3 \times 19}$. We performed experiments with two representations of the pose: all poses aligned to a common referential, and all poses in the camera referential. We estimated the global transformation from the ground-truth. As show in Fig. 7 for all feature types our approach outperforms the baselines. Fig. 8 shows the constraint violation for the different settings. Due to our parameterization, the amount of constraint violation induced by our approach is independent of the pose referential. This is in contrast with the standard GP, which is very sensitive to the representation. In addition, we also enforced the constraints at inference, similarly as [12], but starting from our results. As can be observed from the figures, while this reduced the constraint violation, it had very little influence on the reconstruction error. Fig. 9 depicts some of our results obtained from PHOG features.

### 3.4 Running Time

We compared the running times of our algorithm to those of solving the non-convex constraints at inference [12]. As shown in Table 1, the running times of our algorithm are constant with respect

|  | Constr GP [12] | | | Our approach | | |
|---|---|---|---|---|---|---|
| Training size | 50 | 250 | 500 | 50 | 250 | 500 |
| $2 \times 2$ mesh ($D = 4$) | **2.0** | **5.1** | 21.3 | 8.0 | 7.9 | **8.0** |
| HumanEva ($D = 19$) | 26.1 | 49.6 | 101.0 | **4.9** | **4.9** | **4.8** |
| $9 \times 9$ mesh ($D = 81$) | 1664.9 | 1625.6 | 1599.8 | **8.7** | **8.9** | **9.0** |

Table 1: **Running times comparison.** Average running times per test example in milliseconds for different datasets and different number of training examples. We show results for the constrained GP of [12] and for our approach. Note that, as opposed to [12], our approach is relatively insensitive to the number of training examples and to the dimension of the data.

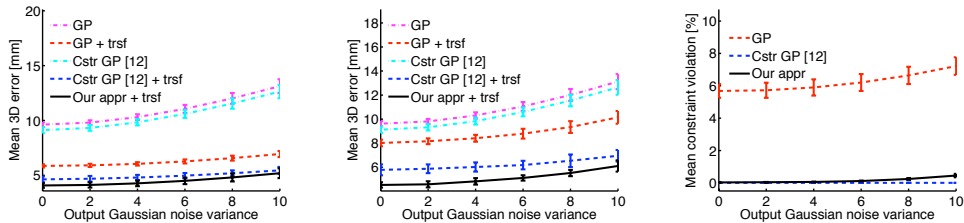

Figure 10: **Robustness to output noise.** Mean reconstruction error and constraint violation as a function of the output noise on the training examples. As a pre-processing step, we projected the noisy training examples to the closest shape that satisfies the constraints. We then trained all approaches with this data. Note that our approach outperforms the baselines.

to the overall size of the problem. This is due to the fact that most of the computation time is spent doing the factorization and not the prediction. In contrast, enforcing constraints at inference is sensitive to the dimension of the data, as well as to the number of training examples[2]. Therefore, for large, high-dimensional training sets, our algorithm is several orders of magnitude faster than [12], and, as shown above, obtains similar or better accuracies.

## 3.5 Robustness to Noise in the Outputs

As shown in Section 2.2, the mean prediction of a GP satisfies linear constraints under the assumption that the training examples all satisfy these constraints. This suggests that our approach might be sensitive to noise on the training outputs, $\mathbf{y}$. To study this, we added Gaussian noise with variance ranging from 2mm to 10mm on the 3D coordinates of the $2 \times 2$ deformable mesh of 100mm side (Fig. 1(b)). To overcome the effect of noise, we first pre-processed the training examples and projected them to the closest shape that satisfies the constraints in a similar manner as in [12]. We then used these rectified shapes as training data for our approach as well as for the baselines. Fig. 10 depicts the reconstruction error and constraint violation as a function of the output noise. We used the image locations of the vertices with noise variance 10 as inputs, and $N = 100$ training examples. Note that our approach outperforms the baselines. Furthermore, our pre-processing step improved the results of all approaches compared to using the original noisy data. Note, however, that in the case of extreme output noise, projecting the training examples on the constraint space might yield meaningless results. This would have a negative impact on the learned predictor, and thus on the performance of all the methods.

## 4 Conclusion

In this paper, we have proposed an approach to implicitly enforcing constraints in discriminative prediction. We have shown that the prediction of a GP always satisfies linear constraints if the training data satisfies these constraints. From this result, we have proposed an effective method to enforce quadratic constraints by changing the parameterization of the problem. We have demonstrated on several rigid and non-rigid monocular pose estimation problems that our method outperforms GP regression, as well as enforcing the constraints at inference [12]. Furthermore, we have shown that our algorithm is very efficient, and makes real-time non-rigid reconstruction an achievable goal. In the future, we intend to investigate other types of image information to estimate the global transformation, as well as study the use of our approach to tasks involving different constraints, such as dynamics.

## Footnotes

[1] In [12], they proposed to optimize either directly the pose, or the vector of kernel values $\mathbf{k}_*$. The second choice requires having more training examples than the number of constraints. Since here this is not always the case, for this dataset we optimized the pose.

[2]For the last dataset, the running times of [12] are independent of $N$. This is due to the fact that, in this case, we optimized the pose directly (see note 1 on page 6).

# References

[1] A. Agarwal and B. Triggs. 3d human pose from silhouettes by relevance vector regression. In *Conference on Computer Vision and Pattern Recognition*, 2004.

[2] M. Alvarez and N. D. Lawrence. Sparse convolved Gaussian processes for multi-output regression. In *Neural Information Processing Systems*, pages 57–64. MIT Press, Cambridge, MA, 2009.

[3] V. Blanz and T. Vetter. A Morphable Model for The Synthesis of 3–D Faces. In *ACM SIGGRAPH*, pages 187–194, Los Angeles, CA, August 1999.

[4] E. Bonilla, K. M. Chai, and C. Williams. Multi-task gaussian process prediction. In J. Platt, D. Koller, Y. Singer, and S. Roweis, editors, *Neural Information Processing Systems*, pages 153–160, Cambridge, MA, 2008. MIT Press.

[5] A. Bosch, A. Zisserman, and X. Munoz. Image classification using random forests and ferns. In *International Conference on Computer Vision*, 2007.

[6] P. Goovaerts. *Geostatistics For Natural Resources Evaluation*. Oxford University Press, 1997.

[7] L. Herda, R. Urtasun, and P. Fua. Hierarchical Implicit Surface Joint Limits to Constrain Video-Based Motion Capture. In *European Conference on Computer Vision*, Prague, Czech Republic, May 2004.

[8] F. Moreno-Noguer, V. Lepetit, and P. Fua. Accurate Non-Iterative $O(n)$ Solution to the P$n$P Problem. In *International Conference on Computer Vision*, Rio, Brazil, October 2007.

[9] M. Perriollat, R. Hartley, and A. Bartoli. Monocular template-based reconstruction of inextensible surfaces. In *British Machine Vision Conference*, 2008.

[10] J. Quinonero-Candela and C. E. Rasmussen. A unifying view of sparse approximate gaussian process regression. *Journal of Machine Learning Research*, pages 1935–1959, 2006.

[11] G. Rogez, J. Rihan, S. Ramalingam, C. Orrite, and P. Torr. Randomized Trees for Human Pose Detection. In *Conference on Computer Vision and Pattern Recognition*, 2008.

[12] M. Salzmann and R. Urtasun. Combining discriminative and generative methods for 3d deformable surface and articulated pose reconstruction. In *Conference on Computer Vision and Pattern Recognition*, San Francisco, CA, June 2010.

[13] M. Salzmann, R. Urtasun, and P. Fua. Local deformation models for monocular 3d shape recovery. In *Conference on Computer Vision and Pattern Recognition*, Anchorage, AK, June 2008.

[14] G. Shakhnarovich, P. Viola, and T. Darrell. Fast pose estimation with parameter-sensitive hashing. In *International Conference on Computer Vision*, Nice, France, 2003.

[15] S. Shen, W. Shi, and Y. Liu. Monocular template-based tracking of inextensible deformable surfaces under l2-norm. In *Asian Conference on Computer Vision*, 2009.

[16] H. Sidenbladh, M. J. Black, and D. J. Fleet. Stochastic Tracking of 3D human Figures using 2D Image Motion. In *European Conference on Computer Vision*, June 2000.

[17] L. Sigal and M. J. Black. Humaneva: Synchronized video and motion capture dataset for evaluation of articulated human motion. Technical Report CS-06-08, Brown University, 2006.

[18] C. Sminchisescu and B. Triggs. Kinematic Jump Processes for Monocular 3D Human Tracking. In *Conference on Computer Vision and Pattern Recognition*, volume I, page 69, Madison, WI, June 2003.

[19] E. Snelson, C. E. Rassmussen and Z. Ghahramani. Warped Gaussian Processes. In *Neural Information Processing Systems*. MIT Press, Cambridge, MA, 2004.

[20] S. Umeyama. Least-squares estimation of transformation parameters between two point patterns. *IEEE Transactions on Pattern Analysis and Machine Intelligence*, 13(4), Apr. 1991.

[21] R. Urtasun and T. Darrell. Sparse Probabilistic Regression for Activity-independent Human Pose Inference. In *Conference on Computer Vision and Pattern Recognition*, Anchorage, AK, 2008.

[22] R. Urtasun, D. Fleet, A. Hertzman, and P. Fua. Priors for people tracking from small training sets. In *International Conference on Computer Vision*, Beijing, China, October 2005.

